# Exact Solutions to Time-Dependent MDPs

**Justin A. Boyan***
ITA Software
Building 400
One Kendall Square
Cambridge, MA 02139
*jab@itasoftware.com*

**Michael L. Littman**
AT&T Labs–Research
and Duke University
180 Park Ave. Room A275
Florham Park, NJ 07932-0971 USA
*mlittman@research.att.com*

## Abstract

We describe an extension of the Markov decision process model in which a continuous time dimension is included in the state space. This allows for the representation and exact solution of a wide range of problems in which transitions or rewards vary over time. We examine problems based on route planning with public transportation and telescope observation scheduling.

## 1 Introduction

Imagine trying to plan a route from home to work that minimizes expected time. One approach is to use a tool such as "Mapquest", which annotates maps with information about estimated driving time, then applies a standard graph-search algorithm to produce a shortest route. Even if driving times are stochastic, the annotations can be expected times, so this presents no additional challenge. However, consider what happens if we would like to include public transportation in our route planning. Buses, trains, and subways vary in their expected travel time *according to the time of day*: buses and subways come more frequently during rush hour; trains leave on or close to scheduled departure times. In fact, even highway driving times vary with time of day, with heavier traffic and longer travel times during rush hour.

To formalize this problem, we require a model that includes both stochastic actions, as in a Markov decision process (MDP), and actions with time-dependent stochastic durations. There are a number of models that include some of these attributes:

- Directed graphs with shortest path algorithms [2]: State transitions are deterministic; action durations are time independent (deterministic or stochastic).
- Stochastic Time Dependent Networks (STDNs) [6]: State transitions are deterministic; action durations are stochastic and can be time dependent.
- Markov decision processes (MDPs) [5]: State transitions are stochastic; action durations are deterministic.
- Semi-Markov decision processes (SMDPs) [5]: State transitions are stochastic; action durations are stochastic, but not time dependent.

In this paper, we introduce the Time-Dependent MDP (TMDP) model, which generalizes all these models by including both stochastic state transitions and stochastic, time-dependent action durations. At a high level, a TMDP is a special continuous-state MDP [5; 4] consisting of states with both a discrete component and a real-valued time component: $\langle x, t \rangle \in X \times \Re$.

With absolute time as part of the state space, we can model a rich set of domain objectives including minimizing expected time, maximizing the probability of making a deadline, or maximizing the dollar reward of a path subject to a time deadline. In fact, using the time dimension to represent other one-dimensional quantities, TMDPs support planning with non-linear utilities [3] (e.g., risk-aversion), or with a continuous resource such as battery life or money.

We define TMDPs and express their Bellman equations in a functional form that gives, at each state $x$, the one-step lookahead value at $\langle x, t \rangle$ for all times in parallel (Section 2). We use the term *time-value function* to denote a mapping from real-valued times to real-valued future reward. With appropriate restrictions on the form of the stochastic state-time transition function and reward function, we guarantee that the optimal time-value function at each state is a piecewise linear function of time, which can be represented exactly and computed by value iteration (Section 3). We conclude with empirical results on two domains (Section 4).

## 2   General model

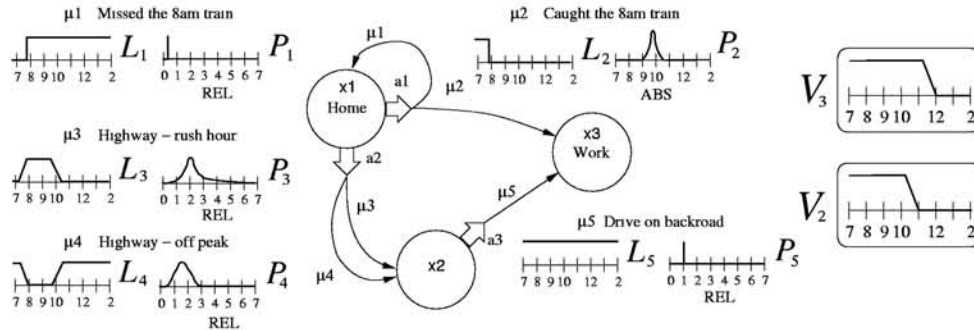

Figure 1: An illustrative route-planning example TMDP.

Figure 1 depicts a small route-planning example that illustrates several distinguishing features of the TMDP model. The start state $x_1$ corresponds to being at home. From here, two actions are available: $a_1$, taking the 8am train (a scheduled action); and $a_2$, driving to work via highway then backroads (may be done at any time).

Action $a_1$ has two possible outcomes, represented by $\mu_1$ and $\mu_2$. Outcome $\mu_1$ ("Missed the 8am train") is active after 7:50am, whereas outcome $\mu_2$ ("Caught the train") is active until 7:50am; this is governed by the likelihood functions $L_1$ and $L_2$ in the model. These outcomes cause deterministic transitions to states $x_1$ and $x_3$, respectively, but take varying amounts of time. Time distributions in a TMDP may be either "relative" (REL) or "absolute" (ABS). In the case of catching the train ($\mu_2$), the distribution is absolute: the arrival time (shown in $P_2$) has mean 9:45am no matter what time before 7:50am the action was initiated. (Boarding the train earlier does not allow us to arrive at our destination earlier!) However, missing the train and returning to $x_1$ has a relative distribution: it deterministically takes 15 minutes from our starting time (distribution $P_1$) to return home.

The outcomes for driving ($a_2$) are $\mu_3$ and $\mu_4$. Outcome $\mu_3$ ("Highway – rush hour") is active with probability 1 during the interval 8am–9am, and with smaller probability outside that interval, as shown by $L_3$. Outcome $\mu_4$ ("Highway – off peak") is complementary. Duration distributions $P_3$ and $P_4$, both relative to the initiation time, show that driving times during rush hour are on average longer than those off peak. State $x_2$ is reached in either case.

From state $x_2$, only one action is available, $a_3$. The corresponding outcome $\mu_5$ ("Drive on backroad") is insensitive to time of day and results in a deterministic transition to state $x_3$ with duration 1 hour. The reward function for arriving at work is +1 before 11am and falls linearly to zero between 11am and noon.

The solution to a TMDP such as this is a *policy* mapping state-time pairs $\langle x, t \rangle$ to actions so as to maximize expected future reward. As is standard in MDP methods, our approach finds this policy via the *value function* $V^*$. We represent the value function of a TMDP as a set of time-value functions, one per state: $V_i(t)$ gives the optimal expected future reward from state $x_i$ at time $t$. In our example of Figure 1, the time-value functions for $x_3$ and $x_2$ are shown as $V_3$ and $V_2$. Because of the deterministic one-hour delay of $\mu_5$, $V_2$ is identical to $V_3$ shifted back one hour. This wholesale shifting of time-value functions is exploited by our solution algorithm.

The TMDP model also allows a notion of "dawdling" in a state. This means the TMDP agent can remain in a state for as long as desired at a reward rate of $K(x, t)$ per unit time before choosing an action. This makes it possible, for example, for an agent to wait at home for rush hour to end before driving to work.

Formally, a TMDP consists of the following components:

$X$    discrete state space

$A$    discrete action space

$M$    discrete set of *outcomes*, each of the form $\mu = \langle x'_\mu, T_\mu, P_\mu \rangle$:

     $x'_\mu \in X$: the resulting state

     $T_\mu \in \{\text{ABS}, \text{REL}\}$: specifies the type of the resulting time distribution

     $P_\mu(t')$ (if $T_\mu = \text{ABS}$): pdf over absolute arrival times of $\mu$

     $P_\mu(\delta)$ (if $T_\mu = \text{REL}$): pdf over durations of $\mu$

$L$    $L(\mu|x, t, a)$ is the likelihood of outcome $\mu$ given state $x$, time $t$, action $a$

$R$    $R(\mu, t, \delta)$ is the reward for outcome $\mu$ at time $t$ with duration $\delta$

$K$    $K(x, t)$ is the reward rate for "dawdling" in state $x$ at time $t$.

We can define the optimal value function for a TMDP in terms of these quantities with the following Bellman equations:

$$V(x, t) = \sup_{t' \geq t} \left( \int_t^{t'} K(x, s)\, ds + \bar{V}(x, t') \right) \quad \text{value function (allowing dawdling)}$$

$$\bar{V}(x, t) = \max_{a \in A} Q(x, t, a) \quad \text{value function (immediate action)}$$

$$Q(x, t, a) = \sum_{\mu \in M} L(\mu|x, a, t) \cdot U(\mu, t) \quad \text{expected Q value over outcomes}$$

$$U(\mu, t) = \begin{cases} \int_{-\infty}^{\infty} P_\mu(t')\ [R(\mu, t, t' - t) + V(x'_\mu, t')]dt' & (\text{if } T_\mu = \text{ABS}) \\ \int_{-\infty}^{\infty} P_\mu(t' - t)[R(\mu, t, t' - t) + V(x'_\mu, t')]dt' & (\text{if } T_\mu = \text{REL}). \end{cases}$$

These equations follow straightforwardly from viewing the TMDP as an undiscounted continuous-time MDP. Note that the calculations of $U(\mu, t)$ are convolutions of the result-time pdf $P$ with the lookahead value $R + V$. In the next section, we discuss a concrete way of representing and manipulating the continuous quantities that appear in these equations.

# 3 Model with piecewise linear value functions

In the general model, the time-value functions for each state can be arbitrarily complex and therefore impossible to represent exactly. In this section, we show how to restrict the model to allow value functions to be manipulated exactly.

For each state, we represent its time-value function $V_i(t)$ as a piecewise linear function of time. $V_i(t)$ is thus represented by a data structure consisting of a set of distinct times called *breakpoints* and, for each pair of consecutive breakpoints, the equation of a line defined over the corresponding interval.

Why are piecewise linear functions an appropriate representation? *Linear* time-value functions provide an exact representation for minimum-time problems. *Piecewise* time-value functions provide closure under the "max" operator.

Rewards must be constrained to be piecewise linear functions of start and arrival times and action durations. We write $R(\mu, t, \delta) = R_s(\mu, t) + R_a(\mu, t + \delta) + R_d(\mu, \delta)$ where $R_s$, $R_a$, and $R_d$ are piecewise linear functions of start time, arrival time, and duration, respectively. In addition, the dawdling reward $K$ and the outcome probability function $L$ must be piecewise constant.

The most significant restriction needed for exact computation is that arrival and duration pdfs be discrete. This ensures closure under convolutions. In contrast, convolving a piecewise constant pdf (e.g., a uniform distribution) with a piecewise linear time-value function would in general produce a piecewise quadratic time-value function; further convolutions increase the degree with each iteration of value iteration. In Section 5 below we discuss how to relax this restriction.

Given the restrictions just mentioned, all the operations used in the Bellman equations from Section 2—namely, addition, multiplication, integration, supremum, maximization, and convolution—can be implemented exactly. The running time of each operation is linear in the representation size of the time-value functions involved. Seeding the process with an initial piecewise linear time-value function, we can carry out value iteration until convergence. In general, the running time from one iteration to the next can increase, as the number of linear "pieces" being manipulated grows; however, the representations grow only as complex as necessary to represent the value function $V$ exactly.

# 4 Experimental domains

We present results on two domains: transportation planning and telescope scheduling. For comparison, we also implemented the natural alternative to the piecewise-linear technique: discretizing the time dimension and solving the problem as a standard MDP. To apply the MDP method, three additional inputs must be specified: an earliest starting time, latest finishing time, and bin width. Since this paper's focus is on exact computations, we chose a discretization level corresponding to the resolution necessary for exact solution by the MDP at its grid points. An advantage of the MDP is that it is by construction acyclic, so it can be solved by just one sweep of standard value iteration, working backwards in time. The TMDP's advantage is that it directly manipulates entire linear segments of the time-value functions.

## 4.1 Transportation planning

Figure 2 illustrates an example TMDP for optimizing a commute from San Francisco to NASA Ames. The 14 discrete states model both location and observed traffic

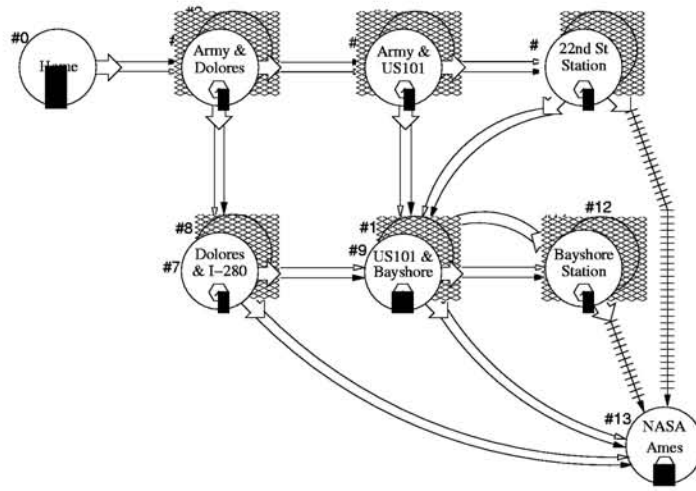

Figure 2: The San Francisco to Ames commuting example

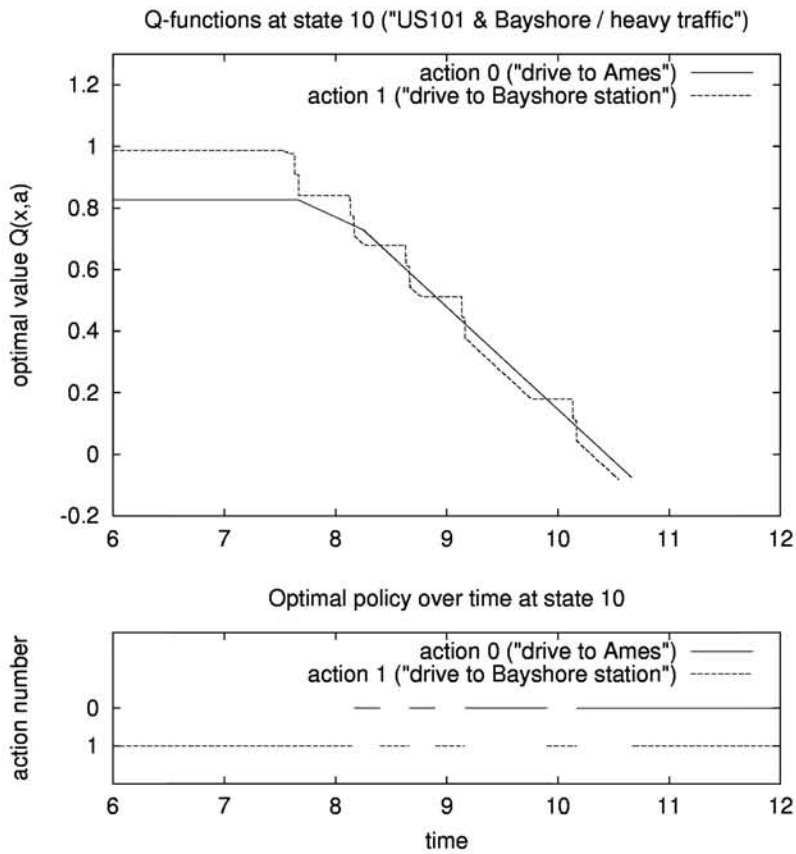

Figure 3: The optimal Q-value functions and policy at state #10.

conditions: shaded and unshaded circles represent heavy and light traffic, respectively. Observed transition times and traffic conditions are stochastic, and depend on both the time and traffic conditions at the originating location. At states 5, 6, 11, and 12, the "catch the train" action induces an absolute arrival distribution reflecting the train schedules.

The domain objective is to arrive at Ames by 9:00am. We impose a linear penalty for arriving between 9 and noon, and an infinite penalty for arriving after noon. There are also linear penalties on the number of minutes spent driving in light traffic, driving in heavy traffic, and riding on the train; the coefficients of these penalties can be adjusted to reflect the commuter's tastes.

Figure 3 presents the optimal time-value functions and policy for state #10, "US101&Bayshore / heavy traffic." There are two actions from this state, corresponding to driving directly to Ames and driving to the train station to wait for the next train. Driving to the train station is preferred (has higher Q-value) at times that are close—but not too close!—to the departure times of the train.

The full domain is solved in well under a second by both solvers (see Table 1). The optimal time-value functions in the solution comprise a total of 651 linear segments.

## 4.2 Telescope observation scheduling

Next, we consider the problem of scheduling astronomical targets for a telescope to maximize the scientific return of one night's viewing [1]. We are given $N$ possible targets with associated coordinates, scientific value, and time window of visibility. Of course, we can view only one target at a time. We assume that the reward of an observation is proportional to the duration of viewing the target. Acquiring a target requires two steps of stochastic duration: moving the telescope, taking time roughly proportional to the distance traveled; and calibrating it on the new target.

Previous approaches have dealt with this stochasticity heuristically, using a just-in-case scheduling approach [1]. Here, we model the stochasticity directly within the TMDP framework. The TMDP has $N+1$ states (corresponding to the $N$ observations and "off") and $N$ actions per state (corresponding to what to observe next). The

| Domain | Solver | Model states | Value sweeps | $V^*$ pieces | Runtime (secs) |
|---|---|---|---|---|---|
| SF-Commute | piecewise VI | 14 | 13 | 651 | 0.2 |
| | exact grid VI | 5054 | 1 | 5054 | 0.1 |
| Telescope-10 | piecewise VI | 11 | 5 | 186 | 0.1 |
| | exact grid VI | 14,311 | 1 | 14,311 | 1.3 |
| Telescope-25 | piecewise VI | 26 | 6 | 716 | 1.8 |
| | exact grid VI | 33,826 | 1 | 33,826 | 7.4 |
| Telescope-50 | piecewise VI | 51 | 6 | 1252 | 6.3 |
| | exact grid VI | 66,351 | 1 | 66,351 | 34.5 |
| Telescope-100 | piecewise VI | 101 | 4 | 2711 | 17.9 |
| | exact grid VI | 131,300 | 1 | 131,300 | 154.1 |

Table 1: Summary of results. The three rightmost columns measure solution complexity in terms of the number of sweeps of value iteration before convergence; the number of distinct "pieces" or values in the optimal value function $V^*$; and the running time. Running times are the median of five runs on an UltraSparc II (296MHz CPU, 256Mb RAM).

dawdling reward rate $K(x, t)$ encodes the scientific value of observing $x$ at time $t$; that value is 0 at times when $x$ is not visible. Relative duration distributions encode the inter-target distances and stochastic calibration times on each transition.

We generated random target lists of sizes $N = 10, 25, 50$, and 100. Visibility windows were constrained to be within a 13-hour night, specified with 0.01-hour precision. Thus, representing the exact solution with a grid required 1301 time bins per state. Table 1 shows comparative results of the piecewise-linear and grid-based solvers.

## 5  Conclusions

In sum, we have presented a new stochastic model for time-dependent MDPs (TMDPs), discussed applications, and shown that dynamic programming with piecewise linear time-value functions can produce optimal policies efficiently. In initial comparisons with the alternative method of discretizing the time dimension, the TMDP approach was empirically faster, used significantly less memory, and solved the problem exactly over continuous $t \in \Re$ rather than just at grid points.

In our exact computation model, the requirement of discrete duration distributions seems particularly restrictive. We are currently investigating a way of using our exact algorithm to generate upper and lower bounds on the optimal solution for the case of *arbitrary* pdfs. This may allow the system to produce an optimal or provably near-optimal policy without having to identify all the twists and turns in the optimal time-value functions. Perhaps the most important advantage of the piecewise linear representation will turn out to be its amenability to bounding and approximation methods. We hope that such advances will enable the solution of city-sized route planning, more realistic telescope scheduling, and other practical time-dependent stochastic problems.

### Acknowledgments

We thank Leslie Kaelbling, Rich Washington and NSF CAREER grant IRI-9702576.

## Footnotes

*The work reported here was done while Boyan's affiliation was with NASA Ames Research Center, Computational Sciences Division.

## References

[1] John Bresina, Mark Drummond, and Keith Swanson. Managing action duration uncertainty with just-in-case scheduling. In *Decision-Theoretic Planning: Papers from the 1994 Spring AAAI Symposium*, pages 19–26, Stanford, CA, 1994. AAAI Press, Menlo Park, California. ISBN 0-929280-70-9. URL http://ic-www.arc.nasa.gov/ic/projects/xfr/jic/jic.html.

[2] Thomas H. Cormen, Charles E. Leiserson, and Ronald L. Rivest. *Introduction to Algorithms*. The MIT Press, Cambridge, MA, 1990.

[3] Sven Koenig and Reid G. Simmons. How to make reactive planners risk-sensitive. In *Proceedings of the 2nd International Conference on Artificial Intelligence Planning Systems*, pages 293–304, 1994.

[4] Harold J. Kushner and Paul G. Dupuis. *Numerical Methods for Stochastic Control Problems in Continuous Time*. Springer-Verlag, New York, 1992.

[5] Martin L. Puterman. *Markov Decision Processes—Discrete Stochastic Dynamic Programming*. John Wiley & Sons, Inc., New York, NY, 1994.

[6] Michael P. Wellman, Kenneth Larson, Matthew Ford, and Peter R. Wurman. Path planning under time-dependent uncertainty. In *Proceedings of the 11th Conference on Uncertainty in Artificial Intelligence*, pages 532–539, 1995.
